# Deterministic Single-Pass Algorithm for LDA

**Issei Sato**
University of Tokyo, Japan
sato@r.dl.itc.u-tokyo.ac.jp

**Kenichi Kurihara**
Google
kenichi.kurihara@gmail.com

**Hiroshi Nakagawa**
University of Tokyo, Japan
n3@dl.itc.u-tokyo.ac.jp

## Abstract

We develop a deterministic single-pass algorithm for latent Dirichlet allocation (LDA) in order to process received documents one at a time and then discard them in an excess text stream. Our algorithm does not need to store old statistics for all data. The proposed algorithm is much faster than a batch algorithm and is comparable to the batch algorithm in terms of perplexity in experiments.

## 1 Introduction

Huge quantities of text data such as news articles and blog posts arrives in a continuous stream. Online learning has attracted a great deal of attention as a useful method for handling this growing quantity of streaming data because it processes data one at a time, whereas batch algorithms are not feasible in these settings because they need all the data at the same time. This paper focus on online learning for Latent Dirichlet allocation (LDA) (Blei et al., 2003), which is a widely used probabilistic model for text data.

Online learning for LDA has been already developed (Banerjee and Basu, 2007; Alsumait et al., 2008; Canini et al., 2009; Yao et al., 2009). Existing studies were based on sampling methods such as the incremental Gibbs sampler and particle filter. Sampling methods seem to be inappropriate for streaming data because sampling methods have to represent a posterior by using a lot of samples, which basically needs much time. Moreover, sampling algorithms often need a resampling

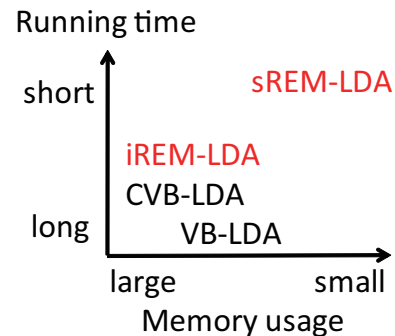

Figure 1: Overview of the relationships among inferences.

step in which a sampling method is applied to old data. Storing old data or old samples adversely affects the good properties of online algorithms. Particle filters also need to run $m$ parallel processing. A parallel algorithm needs more memory than a single-process algorithm, which is not useful for a large quantity of data, especially in the case of a large vocabulary. For example, LDA needs to store the number of words observed in a topic. If the number of topics is $T$, the vocabulary size is $V$ and $m$, so the required memory size is $O(m * T * V)$.

We propose two deterministic online algorithms; an incremental algorithms and a single-pass algorithm. Our incremental algorithm is an incremental variant of the reverse EM (REM) algorithm (Minka, 2001). The incremental algorithm updates parameters by replacing old sufficient statistics with new one for each datum. Our single-pass algorithm is based on an incremental algorithm, but it does not need to store old statistics for all data. In our single-pass algorithm, we propose a sequential update method for the Dirichlet parameters. Asuncion et al. (2009); Wallach et al. (2009) indicated the importance of estimating the parameters of the Dirichlet distribution, which is the distribution over the topic distributions of documents. Moreover, we can deal with the growing vocabulary size. In real life, the total vocabulary size is unknown, i.e., increasing as a document is observed.

In summary, Fig.1 shows the relationships among inferences. VB-LDA is the variational inference for LDA, which is a batch inference; CVB-LDA is the collapsed variational inference for LDA (Teh et al., 2007); iREM-LDA is our incremental algorithm; and sREM-LDA is our single-pass algorithm for LDA.

Sections.2 briefly explains inference algorithms for LDA. Section 3 describes the proposed algorithm for online learning. Section 4 presents the experimental results.

## 2 Overview of Latent Dirichlet Allocation

This section overviews LDA where documents are represented as random mixtures over latent topics and each topic is characterized by a distribution over words. First, we will define the notations, and then, describe the formulation of LDA. $T$ is the number of topics. $M$ is the number of documents. $V$ is the vocabulary size. $N_j$ is the number of words in document $j$. $w_{j,i}$ denotes the $i$-th word in document $j$. $z_{j,i}$ denotes the latent topic of word $w_{j,i}$. $Multi(\cdot)$ is a multinomial distribution. $Dir(\cdot)$ is a Dirichlet distribution. $\boldsymbol{\theta}_j$ denotes a $T$-dimensional probability vector that is the parameters of the multinomial distribution, and represents the topic distribution of document $j$. $\boldsymbol{\beta}_t$ is a multinomial parameter a $V$-dimensional probability where $\beta_{t,v}$ specifies the probability of generating word $v$ given topic $t$. $\boldsymbol{\alpha}$ is the $T$-dimensional parameter vector of the Dirichlet distribution over $\theta_j$ ($j = 1, \cdots, M$).

LDA assumes the following generative process. For each of the $T$ topics $t$, draw $\boldsymbol{\beta}_t \sim Dir(\boldsymbol{\beta}|\lambda) \propto \prod_v \beta_{t,v}^{\lambda-1}$. For each of the $M$ documents $j$, draw $\boldsymbol{\theta}_j \sim Dir(\boldsymbol{\theta}|\boldsymbol{\alpha})$ where $Dir(\boldsymbol{\theta}|\boldsymbol{\alpha}) \propto \prod_t \theta_t^{\alpha_t-1}$.

For each of the $N_j$ words $w_{j,i}$ in document $j$, draw topic $z_{j,i} \sim Multi(z|\boldsymbol{\theta}_j)$ and draw word $w_{j,i} \sim p(w|z_{j,i}, \boldsymbol{\beta})$ where $p(w = v|z = t, \boldsymbol{\beta}) = \beta_{t,v}$.

That is to say, the complete-data likelihood of a document $\boldsymbol{w}_j$ is given by

$$p(\boldsymbol{w}_j, \boldsymbol{z}_j, \boldsymbol{\theta}_j|\boldsymbol{\alpha}, \boldsymbol{\beta}) = p(\boldsymbol{\theta}_j|\boldsymbol{\alpha}) \prod_i^{N_j} p(w_{j,i}|z_{j,i}, \boldsymbol{\beta}) p(\boldsymbol{z}_j|\boldsymbol{\theta}_j). \tag{1}$$

### 2.1 Variational Bayes Inference for LDA

The VB inference for LDA(Blei et al., 2003) introduces a factorized variational posterior $q(\boldsymbol{z}, \boldsymbol{\theta}, \boldsymbol{\beta})$ over $\boldsymbol{z} = \{z_{j,i}\}$, $\boldsymbol{\theta} = \{\boldsymbol{\theta}_j\}$ and $\boldsymbol{\beta} = \{\boldsymbol{\beta}_t\}$ given by

$$q(\boldsymbol{z}, \boldsymbol{\theta}, \boldsymbol{\beta}) = \prod_{j,i} q(z_{j,i}|\boldsymbol{\phi}_{j,i}) \prod_j q(\boldsymbol{\theta}_j|\boldsymbol{\gamma}_j) \prod_t q(\boldsymbol{\beta}_t|\boldsymbol{\mu}_t), \tag{2}$$

where $\boldsymbol{\phi}$ and $\boldsymbol{\gamma}$ are variational parameters, $\phi_{j,i,t}$ specifies the probability that the topic of word $w_{j,k}$ is topic $t$, and $\boldsymbol{\gamma}_j$ and $\boldsymbol{\mu}_t$ are the parameters of the Dirichlet distributions over $\boldsymbol{\theta}_j$ and $\boldsymbol{\beta}_t$, respectively, i.e., $q(\boldsymbol{\theta}_j|\boldsymbol{\gamma}_j) \propto \prod_t \theta_{j,t}^{\gamma_{j,t}-1}$ and $q(\boldsymbol{\beta}_t|\boldsymbol{\mu}_t) \propto \prod_v \beta_{t,v}^{\mu_{t,v}-1}$.

The log-likelihood of documents is lower bounded introducing $q(\boldsymbol{z}, \boldsymbol{\theta})$ by

$$\mathcal{F}[q(\boldsymbol{z}, \boldsymbol{\theta}, \boldsymbol{\beta})] = \int \sum_{\boldsymbol{z}} q(\boldsymbol{z}, \boldsymbol{\theta}, \boldsymbol{\beta}) \log \frac{\prod_j p(\boldsymbol{w}_j, \boldsymbol{z}_j, \boldsymbol{\theta}_j|\boldsymbol{\alpha}, \boldsymbol{\beta}) \prod_t p(\boldsymbol{\beta}_t|\lambda)}{q(\boldsymbol{z}, \boldsymbol{\theta}, \boldsymbol{\beta})} d\boldsymbol{\theta}_j d\boldsymbol{\beta}. \tag{3}$$

The parameters are updated as

$$\phi_{j,i,t} \propto \frac{\exp \Psi(\mu_{t,w_{j,i}})}{\exp \Psi(\sum_v \mu_{t,v})} \exp \Psi(\gamma_{j,t})), \ \gamma_{j,t} = \alpha_t + \sum_{i=1}^{N_j} \phi_{j,i,t}, \ \mu_{t,v} = \lambda + \sum_j n_{j,t,v}, \tag{4}$$

where $n_{j,t,v} = \sum_i \phi_{j,i,t} \mathbb{I}(w_{j,i} = v)$ and $\mathbb{I}(\cdot)$ is an indicator function.

We can estimate $\boldsymbol{\alpha}$ with the fixed point iteration (Minka, 2000; Asuncion et al., 2009) by introducing the gamma prior $G(\alpha_t|a_0, b_0)$, i.e., $\alpha_t \sim G(\alpha_t|a_0, b_0)(t = 1, ..., T)$, as

$$\alpha_t^{new} = \frac{a_0 - 1 + \sum_j \{\Psi(\alpha_t^{old} + n_{j,t}) - \Psi(\alpha_t^{old})\} \alpha_t^{old}}{b_0 + \sum_j (\Psi(N_j + \alpha_0^{old}) - \Psi(\alpha_0^{old}))}, \tag{5}$$

| **Algorithm 1** | **Algorithm 2** |
|---|---|
| VB inference for LDA | CVB inference for LDA |

| | |
|---|---|
| 1: **for** iteration $it = 1, \cdots, L$ **do** | 1: **for** iteration $it = 1, \cdots, L$ **do** |
| 2:    **for** $j = 1, \cdots, M$ **do** | 2:    **for** $j = 1, \cdots, M$ **do** |
| 3:      **for** $i = 1, \cdots, N_j$ **do** | 3:      **for** $i = 1, \cdots, N_j$ **do** |
| 4:        Update $\phi_{j,i,t}$ $(t = 1, \cdots, T)$ by Eq. (4) | 4:        Update $\phi_{j,i,t}$ by Eq. (7) |
| 5:      **end for** | 5:        Update $n_{j,t}$ replacing $\phi_{j,i,t}^{old}$ with $\phi_{j,i,t}^{new}$. |
| 6:      Update $\gamma_{j,t}$ $(t = 1, \cdots, T)$ by Eq. (4) | 6:        Update $n_{t,w_{j,i}}$ replacing $\phi_{j,i,t}^{old}$ with $\phi_{j,i,t}^{new}$. |
| 7:    **end for** | 7:      **end for** |
| 8:    Update $\boldsymbol{\mu}$ by Eq. (4) | 8:    **end for** |
| 9:    Update $\boldsymbol{\alpha}$ by Eq. (5) | 9:    Update $\boldsymbol{\alpha}$ by Eq. (5) |
| 10: **end for** | 10: **end for** |

where $\alpha_0 = \sum_t \alpha_t$, and $a_0$ and $b_0$ are the parameters for the gamma distribution.

Algorithm 1 has the VB inference scheme of LDA.

## 2.2 Collapsed Variational Bayes Inference for LDA

Teh et al. (2007) proposed CVB-LDA inspired by collapsed Gibbs sampling and found that the convergence of CVB-LDA is experimentally faster than that of VB-LDA, and CVB-LDA outperformed VB-LDA in terms of perplexity. The CVB-LDA only introduced a variational posterior $q(\boldsymbol{z})$ where it marginalized out $\boldsymbol{\theta}$ and $\boldsymbol{\beta}$ over the priors. The CVB inference optimizes the following lower bound given by

$$\mathcal{F}_{CVB}[q(\boldsymbol{z})] = \sum_{j=1}^{M} \sum_{\boldsymbol{z}} q(\boldsymbol{z}) \log \frac{p(\boldsymbol{w}_j, \boldsymbol{z}_j | \boldsymbol{\alpha}, \lambda)}{q(\boldsymbol{z})}. \quad (6)$$

The derivation of the update equation for $q(\boldsymbol{z})$ is slightly complicated and involves approximations to compute intractable summations. Although Teh et al. (2007) made use of a second-order Taylor expansion as an approximation, Asuncion et al. (2009) shows the usefulness of an approximation using only zero-order information. An update using only zero-order information is given by

$$\phi_{j,i,t} \propto \frac{\lambda + n_{t,w_{j,i}}^{-j,i}}{V\lambda + \sum_v n_{t,v}^{-j,i}} (\alpha_t + n_{j,t}^{-j,i}), \ n_{j,t} = \sum_{i=1}^{N_j} \phi_{j,i,t}, \ n_{t,v} = \sum_{j,i} \phi_{j,i,t} \mathbb{I}(w_{j,i} = v), \quad (7)$$

where "-j,i" denotes subtracting $\phi_{j,i,t}$. Algorithm 2 provides the CVB inference scheme for LDA.

## 3 Deterministic Online Algorithm for LDA

The purpose of this study is to process text data such as news articles and blog posts arriving in a continuous stream by using LDA. We propose a learning algorithm for LDA that can be applied to these semi-infinite and time-series text streams. For these situations, we want to process text one at a time and then discard them. We repeat iterations only for each word within a document. That is, we update parameters from an arriving document and discard the document after doing $l$ iterations. Therefore, we do not need to store statistics about discarded documents. First, we derived an incremental algorithm for LDA, and then we extended the incremental algorithm to a single-pass algorithm.

### 3.1 Incremental Learning

(Neal and Hinton, 1998) provided a framework of incremental learning for the EM algorithm. In general unsupervised-learning, we estimate sufficient statistics $s_i$ for each data $i$, compute whole

sufficient statistics $\sigma(=\sum_i s_i)$ from all data, and update parameters by using $\sigma$. In incremental learning, for each data $i$, we estimate $s_i$, compute $\sigma^{(i)}$ from $s_i$, and update parameters from $\sigma^{(i)}$. It is easy to extend an existing batch algorithm to the incremental learning if whole sufficient statistics or parameters updates are constructed by simply summarizing all data statistics. The incremental algorithm processes data $i$ by subtracting old $s_i^{old}$ and adding new $s_i^{new}$, i.e., $\sigma^{(i)} = -s_i^{old} + s_i^{new}$. The incremental algorithm needs to store old statistics $\{s_i^{old}\}$ for all data. While batch algorithms update parameters sweeping through all data, the incremental algorithm updates parameters for each data one at a time, which results in more parameter updates than batch algorithms. Therefore, the incremental algorithm sometimes converge faster than batch algorithms.

## 3.2 Incremental Learning for LDA

Our motivation for devising the incremental algorithm for LDA was to compare CVB-LDA and VB-LDA. Statistics $\{n_{t,v}\}$ and $\{n_{j,t}\}$ are updated after each word is updated in CVB-LDA. This update schedule is similar to that of the incremental algorithm. This incremental property seems to be the reason CVB-LDA converges faster than VB-LDA. Moreover, since CVB-LDA optimizes a tighterlower-bound from VB-LDA, CVB-LDA can find better optima. Below, let us consider the incremental algorithm for LDA. We start by optimizing the lower-bound different form VB-LDA by using the reverse EM (REM) algorithm (Minka, 2001) as follows:

$$p(\boldsymbol{w}_j|\boldsymbol{\alpha},\boldsymbol{\beta}) = \int \prod_{i=1}^{N_j} \sum_{t=1}^{T} \prod_{v=1}^{V} (\theta_{j,t}\beta_{t,v})^{\mathbb{I}(w_{j,i}=v)} p(\theta_j|\boldsymbol{\alpha})d\theta_j = \int \prod_{i=1}^{N_j} \sum_{t=1}^{T} (\theta_{j,t}\beta_{t,w_{j,i}})p(\theta_j|\boldsymbol{\alpha})d\theta_j,$$
(8)

$$\geq \int \prod_{i=1}^{N_j} \prod_{t=1}^{T} \left( \frac{\theta_{j,t}\beta_{t,w_{j,i}}}{\phi_{j,i,t}} \right)^{\phi_{j,i,t}} p(\theta_j|\boldsymbol{\alpha})d\theta_j,$$
(9)

$$= \prod_{i=1}^{N_j} \prod_{t=1}^{T} \left( \frac{\beta_{t,w_{j,i}}}{\phi_{j,i,t}} \right)^{\phi_{j,i,t}} \int \prod_{t=1}^{T} \theta_{j,t}^{\sum_i \phi_{j,i,t}} p(\theta_j|\boldsymbol{\alpha})d\theta_j.$$
(10)

Equation (9) is derived from Jensen's inequality as follows. $\log \sum_x f(x) = \log \sum_x q(x)\frac{f(x)}{q(x)} \geq \sum_x q(x)\log\frac{f(x)}{q(x)} = \log \prod_x (\frac{f(x)}{q(x)})^{q(x)}$ where $\sum_x q(x) = 1$, and so $\sum_x f(x) \geq \prod_x (\frac{f(x)}{q(x)})^{q(x)}$.

Therefore, the lower bound for the log-likelihood is given by

$$\hat{\mathcal{F}}[q(\boldsymbol{z})] = \sum_{j,i,t} \phi_{j,i,t} \log \frac{\beta_{t,w_{j,i}}}{\phi_{j,i,t}} + \sum_j \log \left( \frac{\Gamma(\sum_t \alpha_t)}{\Gamma(N_j + \sum_t \alpha_t)} \prod_t \frac{\Gamma(\alpha_t + \sum_i \phi_{j,i,t})}{\Gamma(\alpha_t)} \right).$$
(11)

The maximum of $\hat{\mathcal{F}}[q(\boldsymbol{z})]$ with respect to $q(z_{j,i}=t) = \phi_{j,i,t}$ and $\boldsymbol{\beta}$ is given by

$$\phi_{j,i,t} \propto \beta_{t,w_{j,i}} \exp\{\Psi(\alpha_t + \sum_i \phi_{j,i,t})\}, \ \beta_{tv} \propto \lambda + \sum_j n_{j,t,v},$$
(12)

The updates of $\boldsymbol{\alpha}$ are the same as Eq.(5). Note that we use the maximum a posteriori estiamtion for $\boldsymbol{\beta}$, however, we do not use $\lambda - 1$ to avoid $\lambda - 1 + \sum_j n_{j,t,v}$ taking a negative value.

The lower bound $\hat{\mathcal{F}}[q(\boldsymbol{z})]$ introduces only $q(\boldsymbol{z})$ like CVB-LDA. Equation (12) incrementally updates the topic distribution of a document for each word as in CVB-LDA because we do not need $\gamma_{j,i}$ in Eq.(12) due to marginalizing out of $\boldsymbol{\theta}_j$. Equation (12) is a fixed point update, whereas CVB-LDA can be interpreted as a coordinate ascent algorithm. $\boldsymbol{\alpha}$ and $\boldsymbol{\beta}$ are updated from the entire document. That is, when we compare this algorithm with VB-LDA, it looks like a hybrid variant of a batch updates for $\boldsymbol{\alpha}$ and $\boldsymbol{\beta}$, and incremental updates for $\boldsymbol{\gamma}_j$,

Here, we consider an incremental update for $\boldsymbol{\beta}$ to be analogous to CVBLDA, in which $\boldsymbol{\beta}$ is updated for each word. Note that in the LDA setup, each independent identically distributed data point is a document not a word. Therefore, we incrementally estimate $\boldsymbol{\beta}$ for each document by swapping statistics $n_{j,t,v} = \sum_i^{N_j} \phi_{j,i,t}\mathbb{I}(w_{j,i} = v)$ which is the number of word $v$ generated from topic $t$ in document $j$. Algorithm 3 shows our incremental algorithm for LDA. This algorithm incrementally optimizes the lower bound in Eq.(11).

| **Algorithm 3** | **Algorithm 4** |
|---|---|
| Incremental algorithm for LDA | Single-pass algorithm for LDA |

| | |
|---|---|
| 1: **for** iteration $it = 1, \cdots, L$ **do** | 1: **for** $j = 1, \cdots, M$ **do** |
| 2:    **for** $j = 1, \cdots, M$ **do** | 2:    **for** iteration $it = 1, \cdots, l$ **do** |
| 3:      **for** $i = 1, \cdots, N_j$ **do** | 3:      **for** $i = 1, ..., N_j$ **do** |
| 4:        Update $\phi_{j,i,t}$ by Eq. (12) | 4:        Update $\phi_{j,i,t}$ by Eq. (13). |
| 5:      **end for** | 5:      **end for** |
| 6:      Replace $n_{j,t,v}^{old}$ with $n_{j,t,v}^{new}$ for $v \in$ $\{w_{j,i}\}_{i=1}^{N_j}$ in $\beta$ of Eq. (12) . | 6:      Update $\beta^{(j)}$ by Eq.(13). |
| | 7:      Update $\alpha^{(j)}$ by Eq.(17). |
| 7:    **end for** | 8:    **end for** |
| 8:    Update $\alpha$ by Eq. (5) | 9:    Update $\lambda^{(j)}$ by Eq.(14). |
| 9: **end for** | 10:    Update $\tilde{a}^{(j)}$ and $\tilde{b}^{(j)}$ by Eq.(17). |
| | 11: **end for** |

### 3.3 Single-Pass Algorithm for LDA

Our single-pass algorithm for LDA was inspired by the Bayesian formulation, which internally includes a sequential update. The posterior distribution with the contribution from the data point $x_N$ is separated out so that $p(\theta|\{x_i\}_{i=1}^N) \propto p(x_N|\theta)p(\theta|\{x_i\}_{i=1}^{N-1})$, where $\theta$ denotes a parameter. This indicates that we can use a posterior given an observed datum as a prior for the next datum.. We use parameters learned from observed data as prior parameters for the next data. For example, $\beta_{t,v}$ in Eq. (12) is represented as $\beta_{t,v} \propto \{\lambda + \sum_j^{M-1} n_{j,t,v}\} + n_{M,t,v}$. Here, we can interpret $\{\lambda + \sum_j^{M-1} n_{j,t,v}\}$ as prior parameter $\lambda_{t,v}^{(M-1)}$ for the $M$-th document.

Our single-pass algorithm sequentially sets a prior for each arrived document. By using this sequential setting of prior parameters, we present a single-pass algorithm for LDA as shown in Algorithm 4. First, we update parameters from $j$-th arrived document given prior parameters $\{\lambda_{t,v}^{(j-1)}\}$ for $l$ iterations

$$\phi_{j,i,t} \propto \beta_{t,w_{j,i}}^{(j)} \exp\{\Psi(\alpha_t^{(j)} + \sum_i \phi_{j,i,t})\}, \ \beta_{t,v}^{(j)} \propto \lambda_{t,v}^{(j-1)} + \sum_i^{N_j} \phi_{j,i,t}\mathbb{I}(w_{j,i} = v), \quad (13)$$

where $\lambda_{t,v}^{(0)} = \lambda$ and $\alpha_t^{(j)}$ is explained below. Then, we set prior parameters by using statistics from the document for the next document as follows, and finally discard the document.

$$\lambda_{t,v}^{(j)} = \lambda_{t,v}^{(j-1)} + \sum_i^{N_j} \phi_{j,i,t}\mathbb{I}(w_{j,i} = v). \quad (14)$$

Since the updates are repeated within a document, we need to store statistics $\{\phi_{j,i,t}\}$ for each word in a document, but not for all words in all documents.

In the CVB and iREM algorithms, the Dirichlet parameter, $\alpha$, uses batch updates, i.e., $\alpha$ is updated by using the entire document once in one iteration. We need an online-update algorithm for $\alpha$ to process a streaming text. However, unlike parameter $\beta_{t,v}$, the update of $\alpha$ in Eq.(5) is not constructed by simply summarizing sufficient statistics of data and a prior. Therefore, we derive a single-pass update for the Dirichlet parameter $\alpha$ using the following interpretation.

We consider Eq.(5) to be the expectation of $\alpha_t$ over posterior $G(\alpha_t|\tilde{a}_t, \tilde{b})$ given documents $\mathcal{D}$ and prior $G(\alpha_t|a_0, b_0)$, i.e, $\alpha_t^{new} = \mathbb{E}[\alpha_t]_{G(\alpha|\tilde{a}_t,\tilde{b})} = \dfrac{\tilde{a}_t - 1}{\tilde{b}}$, where

$$\tilde{a}_t = a_0 + \sum_j^M a_{j,t}, \ \tilde{b} = b_0 + \sum_j^M b_j, \quad (15)$$

$$a_{j,t} = \{\Psi(\alpha_t^{old} + n_{j,t}) - \Psi(\alpha_t^{old})\}\alpha_t^{old}, \ b_j = \Psi(N_j + \alpha_0^{old}) - \Psi(\alpha_0^{old}). \quad (16)$$

We regard $a_{j,t}$ and $b_j$ as statistics for each document, which indicates that the parameters that we actually update are $\tilde{a}_t$ and $\tilde{b}$ in Eq.(5). These updates are simple summarizations of $a_{j,t}$ and $b_j$ and prior parameters $a_0$ and $b_0$. Therefore, we have an update for $\alpha_t^{(j)}$ after observing document $j$ given by

$$\alpha_t^{(j)} = \mathbb{E}[\alpha_t]_{G(\alpha|\tilde{a}_t^{(j)},\tilde{b}^{(j)})} = \frac{\tilde{a}_t^{(j)} - 1}{\tilde{b}^{(j)}}, \ \tilde{a}_t^{(j)} = \tilde{a}_t^{(j-1)} + a_{j,t}, \ \tilde{b}^{(j)} = \tilde{b}^{(j-1)} + b_j, \tag{17}$$

$$a_{j,t} = \{\Psi(\alpha_t^{(j-1)} + n_{j,t}) - \Psi(\alpha_t^{(j-1)})\}\alpha_t^{(j-1)}, \ b_j = \Psi(N_j + \alpha_0^{(j-1)}) - \Psi(\alpha_0^{(j-1)}), \tag{18}$$

where $\tilde{a}_t^{(0)} = a_0$ and $\tilde{b}^{(0)} = b_0$.

$\tilde{a}_t^{(j-1)}$ and $\tilde{b}^{(j-1)}$ are used as prior paramters for the next $j$-th documents.

### 3.4  Analysis

This section analyze the proposed updates for parameters $\boldsymbol{\alpha}$ and $\boldsymbol{\beta}$ in the previous section.

We eventually update parameters $\boldsymbol{\alpha}^{(j)}$ and $\boldsymbol{\beta}^{(j)}$ given document $j$ as

$$\alpha_t^{(j)} = \frac{a_0 - 1 + \sum_d^{j-1} a_{d,t} + a_{j,t}}{b_0 + \sum_d^{j-1} b_d + b_j} = \alpha_t^{(j-1)}(1 - \eta_j^{\alpha}) + \eta_j^{\alpha}\frac{a_{j,t}}{b_j}, \ \eta_j^{\alpha} = \frac{b_j}{b_0 + \sum_d^j b_d}. \tag{19}$$

$$\beta_{t,v}^{(j)} = \frac{\lambda + \sum_d^{j-1} n_{d,t,v} + n_{j,t,v}}{V_j\lambda + \sum_d^{j-1} n_{d,t,\cdot} + n_{j,t,\cdot}} = \beta_{t,v}^{(j-1)}(1 - \eta_j^{\beta}) + \eta_j^{\beta}\frac{n_{j,t,v}}{n_{j,t,\cdot}}, \ \eta_j^{\beta} = \frac{(V_j - V_{j-1})\lambda + n_{j,t,\cdot}}{V_j\lambda + \sum_d^j n_{d,t,\cdot}}. \tag{20}$$

where $n_{t,\cdot} = \sum_v n_{t,v}$ and $V_j$ is the vocabulary size of total observed documents($d = 1, \cdots, j$). Our single-pass algorithm sequentially sets a prior for each arrived document, and so we can select a prior (a dimension of Dirichlet distribution) corresponding to observed vocabulary. In fact, this property is useful for our problem because the vocabulary size is growing in the text stream. These updates indicate that $\eta_j^{\alpha}$ and $\eta_j^{\beta}$ interpolate the parameters estimated from old and new data. These updates look like a stepwise algorithm (H.Robbins and S.Monro, 1951; Sato and Ishii, 2000), although a stepsize algorithm interpolates sufficient statistics whereas our updates interpolate parameters. In our updates, how we set the stepsize for parameter updates is equivalent to how we set the hyper-parameters for priors. Therefore, we do not need to newly introduce a stepsize parameter.

In our update of $\boldsymbol{\beta}$, the appearance rate of word $v$ in topic $t$ in document $j$, $n_{j,t,v}/n_{j,t,\cdot}$, is added to old parameter $\beta_{t,v}^{(j-1)}$ with weight $\eta_j^{\beta}$, which gradually decreases as the document is observed. The same relation holds for $\boldsymbol{\alpha}$. Therefore, the influence of new data decreases as the number of document observations increases as shown in Theorem 1. Moreover, Theorem 1 is an important role in analyzing the convergence of parameter updates by using the super-martingale convergence theorem (Bertsekas and Tsitsiklis, 1996; Brochu et al., 2004). This convergence analysis is our future work.

**Theorem 1.** *If $\epsilon$ and $\nu$ exist satisfying $0 < \epsilon < S_j < \nu$ for any $j$,*

$$\eta_j = \frac{S_j}{\tau + \sum_d^j S_d} \tag{21}$$

*satisfies*

$$\lim_{j \to \infty} \eta_j = 0, \ \sum_j^{\infty} \eta_j = \infty, \ \sum_j^{\infty} \eta_j^2 < \infty \tag{22}$$

Note that $\eta_j^{\alpha}$ and $\eta_j^{\beta}$ are shown as $\eta_j$ given by Eq. (21). The proof is given in the supporting material.

# 4 Experiments

We carried out experiments on document modeling in terms of perplexity. We compared the inferences for LDA in two sets of text data. The first was "Associated Press(AP)" where the number of documents was $M = 10,000$ and the vocabulary size was $V = 67,291$. The second was "The Wall Street Journal(WSJ)" where $M = 10,000$ and $V = 56,738$. The ordering of document is time-series. The comparison metric for document modeling was the "test set perplexity". We randomly split both data sets into a training set and a test set by assigninig $20\%$ of the words in each document to the test set. Stop words were eliminated in datasets.

We performed experiments on six inferences, PF, VB, CVB0, CVB, iREM and sREM. PF denotes the particle filter for LDA used in Canini et al. (2009). We set $\alpha_t$ as $50/T$ in PF. The number of particles, denoted by $P$, is $64$. The number of words for resampling, denoted by $R$, is $20$. The effective sample size (ESS) threshold, which controls the number of resamplings, is set at 10. CVB0 and CVB are collapsed variational inference for LDA using zero-order and second-order information, respectively. iREM represents the incremental reverse EM algorithm in Algorithm 3. CVB0 and CVB estimates the Dirichlet parameter $\alpha$ over the topic distribution for all datasets, i.e., a batch framework. We estimated $\alpha$ in iREM for all datasets like CVB to clarify the properties of iREM compared with CVB. $L$ denotes the number of iterations for whole documents in Algorithms 1 and 2. sREM indicates a single-pass variants of iREM in Algorithm 4. $l$ denotes the number of iterations within a document in Algorithm 4. sREM does not make iterations for whole documents.

Figure 2 demonstrates the results of experiments on the test set perplexity where lower values indicates better performance. We ran experiments five times with different random initializations and show the averages [1]. PF and sREM calculate the test set perplexity after sweeping through all traing set.

VB converges slower than CVB and iREM. Moreover, iREM outperforms CVB in the convergence rate. Although CVB0 outperforms other algorithms for the cases of low number of topics, the convergence rate of CVB0 depends on the number of topics. sREM does not outperform iREM in terms of perplexities, however, the performance of sREM is close to that of iREM
As a results, we recommend sREM in a large number of documents or document streams. sREM does not need to store old statistics for all documents unlike other algorithms. In addition, the convergence of sREM depends on the length of a document, rather than the number of documents. Since we process each document individually, we can control the number of iterations corresponding to the length of each arrived document. Finally, we discuss the running time. The running time of sREM is $O(\frac{L}{l})$ times shorter than that of VB, CVB0, CVB and iREM. The averaged running times of PF(T=300,P=64,R=20) are 28.2 hours in AP and 31.2 hours in WSJ. Those of sREM(T=300,l=5) are 1.2 hours in AP and 1.3 hours in WSJ.

# 5 Conclusions

We developed a deterministic online-learning algorithm for latent Dirichlet allocation (LDA). The proposed algorithm can be applied to excess text data in a continuous stream because it processes received documents one at a time and then discard them. The proposed algorithm was much faster than a batch algorithm and was comparable to the batch algorithm in terms of perplexity in experiments.

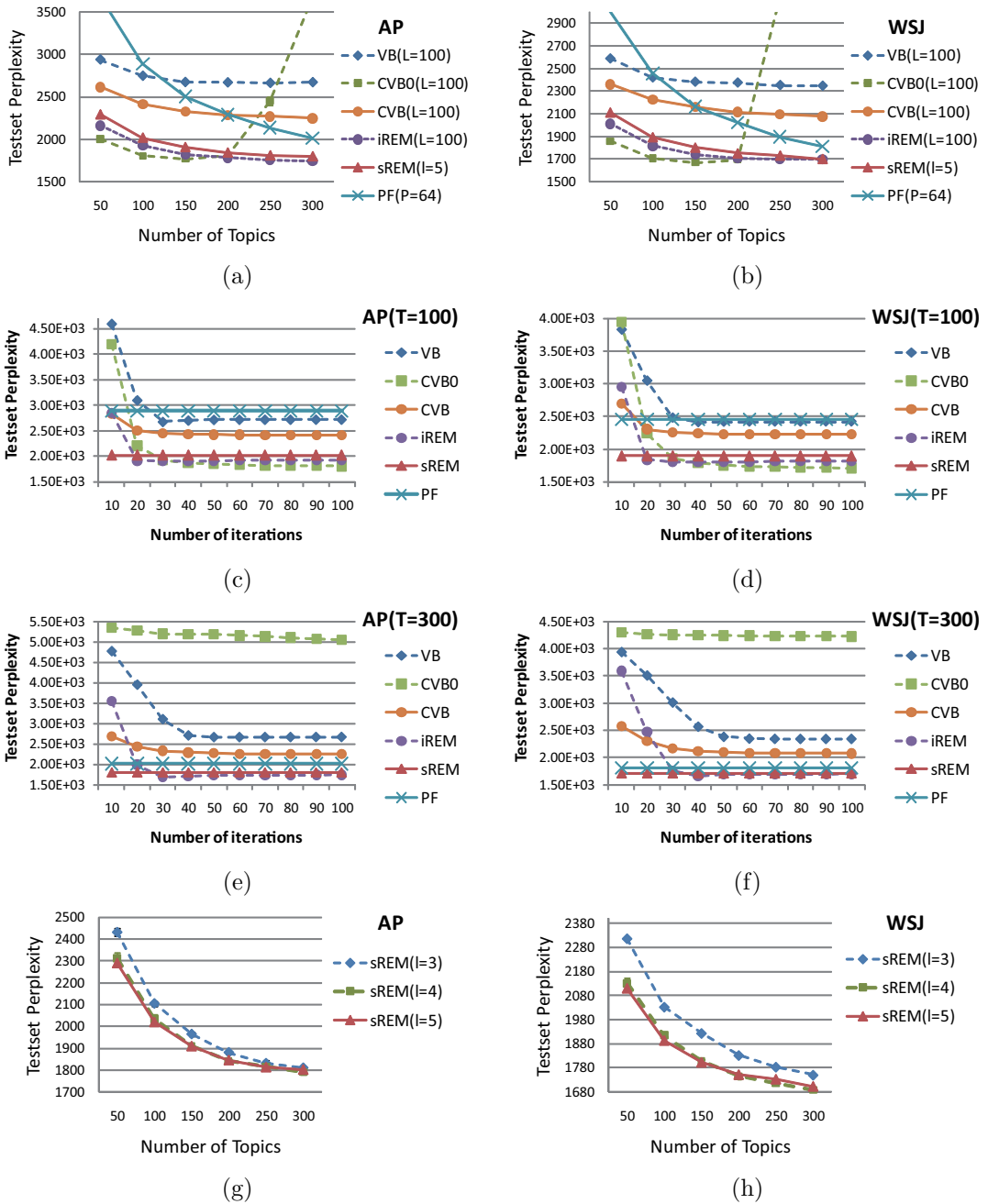

Figure 2: Results of experiments. Left line indicates the results in AP corpus. Right line indicates the results in WSJ corpus. (a) and (b) compared test set perplexity with respect to the number of topics. (c), (d), (e) and (f) compared test set perplexity with respect to the number of iterations in topic $T = 100$ and $T = 300$, respectively. (g) and (h) show the relationships between test set perplexity and the number of iterations within a document, i.e., $l$.

## Footnotes

[1]We exclude the error bar with standard deviation because it is so small that it is hidden by the plot markers

# References

Loulwah Alsumait, Daniel Barbara, and Carlotta Domeniconi. On-line lda: Adaptive topic models for mining text streams with applications to topic detection and tracking. *IEEE International Conference on Data Mining*, 0:3–12, 2008. ISSN 1550-4786.

A. Asuncion, M. Welling, P. Smyth, and Y. W. Teh. On smoothing and inference for topic models. In *Proceedings of the International Conference on Uncertainty in Artificial Intelligence*, 2009.

Arindam Banerjee and Sugato Basu. Topic models over text streams: A study of batch and online unsupervised learning. In *SIAM International Conference on Data Mining*, 2007.

D. P. Bertsekas and J. N. Tsitsiklis. *Neuro-Dynamic Programming*. Athena Scientific, 1996.

D. M. Blei, A. Y. Ng, and M. I. Jordan. Latent Dirichlet allocation. *Journal of Machine Learning Research*, 3:993–1022, 2003.

Eric Brochu, Nando de Freitas, and Kejie Bao. Owed to a martingale: A fast bayesian on-line em algorithm for multinomial models, 2004.

Kevin R. Canini, Lei Shi, and Thomas L. Griffiths. Online inference of topics with latent dirichlet allocation. In *Proceedings of the Twelfth International Conference on Artificial Intelligence and Statistics*, 2009.

H.Robbins and S.Monro. A stochastic approximation method. In *Annals of Mathematical Statistics*, pages 400–407, 1951.

Thomas P. Minka. Estimating a dirichlet distribution. Technical report, Microsoft, 2000. URL `http://research.microsoft.com/~minka/papers/dirichlet/minka-dirichlet.pdf`.

Thomas P. Minka. Using lower bounds to approximate integrals. Technical report, Microsoft, 2001. URL `http://research.microsoft.com/en-us/um/people/minka/papers/rem.html`.

R. Neal and G. Hinton. A view of the EM algorithm that justifies incremental, sparse, and other variants. In M. I. Jordan, editor, *Learning in Graphical Models*. Kluwer, 1998. URL `http://citeseerx.ist.psu.edu/viewdoc/summary?doi=10.1.1.33.2557`.

Masa A. Sato and Shin Ishii. On-line em algorithm for the normalized gaussian network. *Neural Computation*, 12(2):407–432, 2000. URL `http://citeseerx.ist.psu.edu/viewdoc/summary?doi=10.1.1.37.3704`.

Yee Whye Teh, David Newman, and Max Welling. A collapsed variational Bayesian inference algorithm for latent Dirichlet allocation. In *Advances in Neural Information Processing Systems 19*, 2007.

Hanna Wallach, David Mimno, and Andrew McCallum. Rethinking lda: Why priors matter. In Y. Bengio, D. Schuurmans, J. Lafferty, C. K. I. Williams, and A. Culotta, editors, *Advances in Neural Information Processing Systems 22*, pages 1973–1981. 2009.

Limin Yao, David Mimno, and Andrew McCallum. Efficient methods for topic model inference on streaming document collections. In *KDD '09: Proceedings of the 15th ACM SIGKDD international conference on Knowledge discovery and data mining*, pages 937–946, New York, NY, USA, 2009. ACM. ISBN 978-1-60558-495-9.

